# A Neural Network Model of Naive Preference and Filial Imprinting in the Domestic Chick

**Lucy E. Hadden**
Department of Cognitive Science
University of California, San Diego
La Jolla, CA 92093
hadden@cogsci.ucsd.edu

## Abstract

Filial imprinting in domestic chicks is of interest in psychology, biology, and computational modeling because it exemplifies simple, rapid, innately programmed learning which is biased toward learning about some objects. Horn et al. have recently discovered a naive visual preference for heads and necks which develops over the course of the first three days of life. The neurological basis of this predisposition is almost entirely unknown; that of imprinting-related learning is fairly clear. This project is the first model of the predisposition consistent with what is known about learning in imprinting. The model develops the predisposition appropriately, learns to "approach" a training object, and replicates one interaction between the two processes. Future work will replicate more interactions between imprinting and the predisposition in chicks, and analyze why the system works.

## 1 Background

Filial imprinting in domestic chicks is of interest in psychology, biology, and computational modeling (O'Reilly and Johnson, 1994; Bateson and Horn, 1994) because it exemplifies simple, rapid, innately programmed learning which is biased toward learning about some particular objects, and because it has a sensitive period in which learning is most efficient. Domestic chicks will imprint on almost anything (including boxes, chickens, and humans) which they see for enough time (Horn, 1985). Horn and his colleagues (Horn, 1985) have recently found a naive visual preference (predisposition) for heads and necks which develops over the course of the first three days of life. In particular, the birds prefer to approach objects shaped like heads and necks, even if they are the heads and necks of other species, including ducks and polecats (Horn, 1985). This preference interacts interestingly with filial imprinting, or learning to recognize a parent. Chicks can still learn about (and imprint

on) other objects even in the presence of this predisposition, and the predisposition can override previously learned preferences (Johnson et al., 1985), which is usually hard with imprinted chicks. These interactions are like other systems which rely on naive preferences and learning.

While the neurological basis of imprinting is understood to some extent, that of the predisposition for heads and necks is only beginning to be investigated. Imprinting learning is known to take place in IMHV (intermediate and medial portions of the hyperstriatum ventrale) (Horn, 1985), and to rely on noradrenaline (Davies et al., 1992). The predisposition's location is currently unknown, but its strength correlates with plasma testosterone levels (Horn, 1985).

## 1.1   Previous Models

No previous models of imprinting have incorporated the predisposition in any meaningful way. O'Reilly & Johnson's (1994) model focussed on accounting for the sensitive period via an interaction between hysteresis (slow decay of activation) and a Hebbian learning rule, and ignored the predisposition. The only model which did try to include a predisposition (Bateson and Horn, 1994) was a 3-layer Hebbian network with real-valued input vectors, and outputs which represented the strength of an "approach" behavior. Bateson and Horn (1994) found a "predisposition" in their model by comparing networks trained on input vectors of 0s and 1s (High) to vectors where non-zero entries were 0.6 (Low). Untrained networks preferred (produced a higher output value for) the high-valued input ("hen"), and trained networks preferred the stimulus they were trained on ("box"). Of course, in a network with identical weights, an input with higher input values will naturally excite an output unit more than one with lower input values. Thus, this model's predisposition is implicit in the input values, and is therefore hard to apply to chicks.

In this project, I develop a model which incorporates both the predisposition and imprinting, and which is as consistent as possible with the known neurobiology. The overall goals of the project are to clarify how this predisposition might be implemented, and to examine more generally the kinds of representations that underlie naive preferences that interact with and facilitate, rather than replace, learning. These particular simulations show that the model exhibits the same qualitative behavior as chicks under three important sets of conditions.

The rest of the paper first describes the architecture of the current model (in general terms and then in more detail). It goes on to describe the particular simulations, and then compares the results of those simulations with the data gathered from chicks.

## 2   Architecture

The neural network model's architecture is shown in Figure 1. The input layer is a 6x6 pixel "retina" to which binary pictures are presented. The next layer is a feature detector. The predisposition serves as the home of the network's naive preference, while the IMLL (intermediate learning layer) is intended to correspond to a chick's IMHV, and is where the network stores its learned representations. The output layer consists of two units which are taken to represent different action patterns (following Bateson and Horn (1994)): an "approach" unit and a "withdraw" unit. These are the two chick behaviors which researchers use to assess a chick's degree of preference for a particular stimulus. The feature detector provides input to the predisposition and IMLL layers; they in turn provide input to the output layer. Where there are connections, layers (and subparts) are fully interconnected.

The feature detector uses a linear activation function; the rest of the network has a hyperbolic tangent activation function. All activations and all connections can be either positive

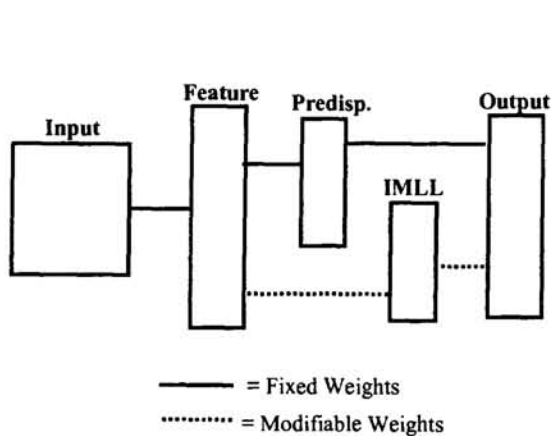

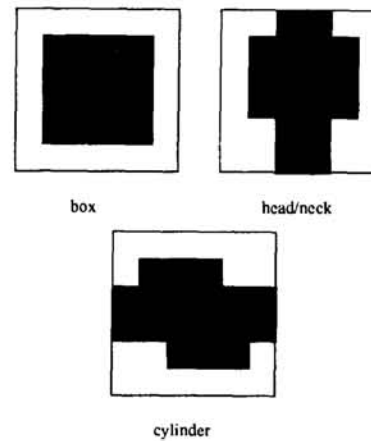

Figure 1: The network architecture sketched. All connections are feedforward (from input toward output) only.

Figure 2: The three input patterns used by the network. They have between 16 and 18 pixels each, and the central moment of each image is the same.

or negative; the connections are limited to ±0.9. Most of the learning takes place via a covariant Hebb rule, because it is considered to be plausible neurally.

The lowest level of the network is a feature-detecting preprocessor. The current implementation of this network takes crude 6x6 binary pictures (examples of which can be seen in Fig. 2), and produces a 15-place floating-point vector. The output units of the feature detector are clustered into five groups of three units each; each group of three units operates under a winner-take-all rule, in order to increase the difference between preprocessed patterns for the relevant pictures. The feature detector was trained on random inputs for 400 cycles with a learning rate of .01, and its weights were then frozen. Training on random input was motivated by the finding that the lower levels of the visual system require some kind of input in order to organize; Miller et al. (1989) suggest that, at least in cats, the random firing of retinal neurons is sufficient.

The predisposition layer was trained via backprop using the outputs of the feature detector as its inputs. The pattern produced by the "head-neck" picture in the feature detector was trained to excite the "approach" output unit and inhibit the "withdraw" unit; other patterns were trained to a neutral value on both output units. These weights were stored, and treated as fixed in the larger network. (In fact, these weights were scaled down by a constant factor (.8) before being used in the larger network.) Since this is a naive preference, or predisposition, these weights are assumed to be fixed evolutionarily. Thus, the method of setting them is irrelevant; they could also have been found by a genetic algorithm.

The IMLL layer is a winner-take-all network of three units. Its connections with the feature detector's outputs are learned by a Hebb rule with learning rate .01 and a weight decay (to 0) term of .0005. For these simulations, its initial weights were fixed by hand, in a pattern which insured that each IMLL unit received a substantially different value for the same input pattern. This pattern of initial weights also increased the likelihood that the three patterns of interest in the simulations maximally affected different IMLL units.

As previously mentioned, the output layer consists of an "approach" and a "withdraw" unit. It also learns via a Hebb rule, with the same learning rate and decay term as IMLL. Its connections with IMLL are learned; those with the predisposition layer are fixed. Initial weights between IMLL and the output layer are random, and vary from -0.3 to 0.3. The bias to the approach unit is 0; that to the withdraw unit is 0.05.

## 2.1 Training

In the animal experiments on which this model is based, chicks are kept in the dark (and in isolation) except for training and testing periods. Training periods involve visual exposure to an object (usually a red box); testing involves allowing the chick to choose between approaching the training object and some other object (usually either a stuffed hen or a blue cylinder) (Horn, 1985). The percentage of time the chick approaches the training object (or other object of interest) is its preference score for that object (Horn, 1985). A preference score of 50% indicates indifference; scores above 50% indicate a preference for the target object, and those below indicate a preference for the other object. For the purposes of modeling, the most relevant information is the change (particularly the direction of change) in the preference score between two conditions.

Following this approach, the simulations use three preset pictures. One, a box, is the only one for which weights are changed; it is the training pattern. The other two pictures are test patterns; when they are shown, the network's weights are not altered. One of these test patterns is the head/neck picture on which the predisposition network was trained; the other is a cylinder. As with chicks, the behavioral measure is the preference score. For the network, this is calculated as pref. score $= 100 \times a_t/(a_t + a_c)$, where $a_t$ is the activation of the approach unit when the network is presented with the training (or target) picture, and $a_c$ is the activation of the approach unit given the comparison picture. It is assumed that both values are positive; otherwise, the approach unit is taken to be off.

In these simulations, the network gets the training pattern (a "box") during training periods, and random input patterns (simulating the random firing of retinal neurons) otherwise. The onset of the predisposition is modeled by allowing the predisposition layer to help activate the outputs only after the network receives an "experience" signal. This signal models the sharp rise in plasma testosterone levels in dark-reared chicks following any sort of handling (Horn, 1985). Once the network has received the "experience" signal, the weights are modified for random input as well as for the box picture. Until then, weights are modified only for the box picture. Real chicks can be tested only once because of the danger of one-trial learning, so all chick data compares the behavior of groups of chicks under different conditions. The network's weights can be kept constant during testing, and the same network's responses can be measured before and after it is exposed to the relevant condition. All simulations were 100 iterations long.

## 3 Simulations

The simulations using this model currently address three phenomena which have been studied in chicks. First, in simple imprinting chicks learn to recognize a training object, and usually withdraw from other objects once they have imprinted on the training object. This simulation requires simply exposing the network to the training object and measuring its responses. The model "imprints" on the box if its preference for the box relative to both the head/neck and cylinder pictures increases during training. Ideally, the value of the approach unit for the cylinder and box will also decrease, to indicate the network's tendency to withdraw from "unfamiliar" stimuli.

Second, chicks with only the most minimal experience (such as being placed in a dark running wheel) develop a preference for a stuffed fowl over other stimuli. That is, they will approach the fowl significantly more than another object (Horn, 1985). This is modeled by turning on the "predisposition" and allowing the network to develop with no training whatsoever. The network mimics chick behavior if the preference score for the head/neck picture increases relative to the box and the cylinder pictures.

Third, after the predisposition has been allowed to develop, training on a red box decreases

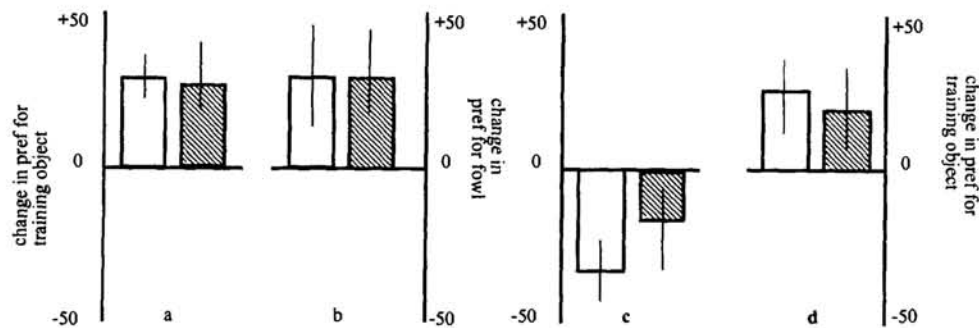

Figure 3: A summary of the results of the model. All bars are differences in preference scores between conditions for chicks (open bars) and the model (striped bars). a: Imprinting (change in preference for training object): trained − untrained. b: Predisposition (change in preference for fowl): experience − no experience (predisposition − no predisposition). c: Change in preference for fowl vs. box: trained − predisposition only. d: Change in preference for box vs. cylinder: trained − predisposition only. (Chick data adapted from (Horn, 1985; Bolhuis et al., 1989).)

a chick's preference for the fowl relative to the box. It also increases the chick's preference for the box relative to a blue cylinder or other unfamiliar object (Bolhuis et al., 1989). In the model, the predisposition layer is allowed to activate the output layer for 20 iterations before training starts. Then the model is exposed to the network for 25 iterations. If its preference score for the fowl decreases after training, the network has shown the same pattern as chicks.

## 4  Results and Discussion

A summary of the results is shown in Figure 3. Since these simulations try to capture the qualitative behavior of chicks, all results are shown as the change in preference scores between two conditions. For the chick data, the changes are approximate, and calculated from the means only. The network data is the average of the results for 10 networks, each with a different random seed (and therefore initial weight patterns). For the three conditions tested, the model's preference scores moved in the same direction as the chicks.

The interaction between imprinting and the predisposition cannot be investigated computationally unless the model displays both behaviors. These baseline behaviors are shown in Fig. 3-a and b. Trained chicks prefer the training object more after training than before (Horn, 1985); so does the model (Fig. 3-a). In the case of the predisposition (Fig. 3-b), the bird data is a difference between preferences for a stuffed fowl in chicks which had developed the predisposition (and therefore preferred the fowl) and those which had not (and therefore did not). Similarly, the network preferred the head/neck picture more after the predisposition had been allowed to develop than at the beginning of the simulation.

The interactions between imprinting and the predisposition are the real measures of the model's success. In Fig. 3-c, the predisposition has been allowed to develop before training begins. Trained birds with the predisposition were compared with untrained birds also with the predisposition (trained − untrained). Trained birds preferred the stuffed fowl less than their untrained counterparts (Bolhuis et al., 1989). The network's preference score just before training is subtracted from its score after training. As with the real chicks, the network prefers the head/neck picture less after training than it did before. Fig. 3-d shows that, as with chicks, the network's preference for the box increased relative to that for the cylinder during the course of training. For these three conditions, then, the model is

qualitatively a success.

## 4.1   Discussion of Basic Results

The point of network models is that their behavior can be analyzed and understood more easily than animals'. The predisposition's behavior is quite simple: to the extent that a random input pattern is similar to the head/neck picture, it activates the predisposition layer, and through it the approach unit. Thus the winning unit in IMLL is correlated with the approach unit, and the connections are strengthened by the Hebb rule. Imprinting is similar, but only goes through the IMLL layer, so the approach unit may not be on. In both cases, the weights from the other units decay slowly during training, so that usually the other input patterns fail to excite the approach unit, and even excite the withdraw unit slightly because of its small positive bias. Only one process is required to obtain both the predisposition and imprinting, since both build representations in IMLL.

The interaction between imprinting and the predisposition first increases the preference for the predisposition, and then alters the weights affecting the reaction to the box picture. The training phase acts just like the ordinary imprinting phase, so that preference for both the head/neck and the cylinder decrease during training.

Some exploration of the relevant parameters suggests that the predisposition's behavior does not depend simply on its strength. Because IMLL is a winner-take-all layer, changing the predisposition's strength can, by moving the winning node around during training, cause previous learning to be lost. Such motion obviously has a large effect on the outcome of the simulation.

## 4.2   Temporal aspects of imprinting

The primary weakness of the model is its failure to account for some critical temporal aspects of imprinting. It is premature to draw many conclusions about chicks from this model, because it fails to account for either long-term sensitive periods or the short-term time course of the predisposition.

Neither the predisposition nor imprinting in the model have yet been shown to have sensitive periods, though both do in real birds (Horn, 1985; Johnson et al., 1985). Preliminary results, however, suggest that imprinting in the networks does have a sensitive period, presumably because of weight saturation during learning. It is not yet clear whether the predisposition's sensitive period will require an exogenous process.

Second, the model does not yet show the appropriate time course for the development of the predisposition. In chicks, the predisposition develops fairly slowly over the course of five or so hours (Johnson et al., 1985). In chicks for which the first experience is training, the predisposition's effect is to increase the bird's preference for the fowl regardless of training object, over the course of the hours following training (Johnson et al., 1985). In the model, the predisposition appears quickly and, because of weight decay and other factors, the strength of the predisposition slowly decreases over the iterations following training, rather than increasing. Increasing the learning rate of IMLL over time could solve this problem. Once it exhibits time course behaviors, especially if no further processes need to be postulated, the model will facilitate interesting analyses of how a simple set of processes and assumptions can interact to produce highly complicated behavior.

## 5   Conclusion

This model displays some important interactions between learning and a predisposition in filial imprinting. It is the first which accounts for the predisposition at all. Other models

of imprinting have either ignored the issue or built in the predisposition by hand. In this model, the interaction between two simple systems, a fixed predisposition and a learned approach system, gives rise to one important more complex behavior. In addition, the two representations of the head/neck predisposition can account for lesion studies in which lesioning IMLL removes a chick's memory of its training object or prevents it from learning anything new about specific objects, but leaves the preference for heads and necks intact (Horn, 1985). Clearly, if the IMLL layer is missing, the network loses any information it might have learned about training objects, and is unable to learn anything new from future training. The predisposition, however, is still intact and able to influence the network's behavior.

The nature of predispositions like chicks' naive preference for heads and necks, and how they interact with learning, are interesting in a number of fields. Morton and Johnson (1991) have already explored the similarities between chicks' preferences for heads and necks and human infants' preferences for human faces. Such naive preferences are also important in any discussion of innate information, and the number of processes needed to handle innate and learned information. Although this model and its successors cannot directly address these issues, I hope that their explication of how fairly general predispositions can influence learning will improve understanding of some of the mechanisms underlying them.

## Acknowledgements

This work was supported by a fellowship from the National Physical Sciences Consortium.

# References

P. Bateson and G. Horn. Imprinting and recognition memory: A neural net model. *Animal Behaviour*, 48(3):695–715, 1994.

J. J. Bolhuis, M. H. Johnson, and G. Horn. Interacting mechanisms during the formation of filial preferences: The development of a predisposition does not prevent learning. *Journal of Experimental Psychology: Animal Behavior Processes*, 15(4):376–382, 1989.

D. C. Davies, M. H. Johnson, and G. Horn. The effect of the neurotoxin dsp4 on the development of a predisposition in the domestic chick. *Developmental Psychobiology*, 25(2):251–259, 1992.

G. Horn. *Memory, Imprinting, and the Brain: An inquiry into mechanisms*. Clarendon Press, Oxford, 1985.

M. H. Johnson, J. J. Bolhuis, and G. Horn. Interaction between acquired preferences and developing predispositions during imprinting. *Animal Behaviour*, 33(3):1000–1006, 1985.

K. Miller, J. Keller, and M. Stryker. Ocular dominance column development: analysis and simulation. *Science*, 245:605–615, 1989.

J. Morton and M. H. Johnson. Conspec and conlern: a two-process theory of infant face recognition. *Psychological Review*, 98(2):164–181, 1991.

R. C. O'Reilly and M. H. Johnson. Object recognition and sensitive periods: A computational analysis of visual imprinting. *Neural Computation*, 6(3):357–389, 1994.